# Beyond maximum likelihood and density estimation: A sample-based criterion for unsupervised learning of complex models

**Sepp Hochreiter and Michael C. Mozer**
Department of Computer Science
University of Colorado
Boulder, CO 80309–0430
{hochreit,mozer}@cs.colorado.edu

## Abstract

The goal of many unsupervised learning procedures is to bring two probability distributions into alignment. Generative models such as Gaussian mixtures and Boltzmann machines can be cast in this light, as can recoding models such as ICA and projection pursuit. We propose a novel sample-based error measure for these classes of models, which applies even in situations where maximum likelihood (ML) and probability density estimation-based formulations cannot be applied, e.g., models that are nonlinear or have intractable posteriors. Furthermore, our sample-based error measure avoids the difficulties of approximating a density function. We prove that with an unconstrained model, (1) our approach converges on the correct solution as the number of samples goes to infinity, and (2) the expected solution of our approach in the generative framework is the ML solution. Finally, we evaluate our approach via simulations of linear and nonlinear models on mixture of Gaussians and ICA problems. The experiments show the broad applicability and generality of our approach.

## 1  Introduction

Many unsupervised learning procedures can be viewed as trying to bring two probability distributions into alignment. Two well known classes of unsupervised procedures that can be cast in this manner are *generative* and *recoding* models. In a generative unsupervised framework, the environment generates training examples—which we will refer to as *observations*—by sampling from one distribution; the other distribution is embodied in the model. Examples of generative frameworks are mixtures of Gaussians (*MoG*) [2], factor analysis [4], and Boltzmann machines [8]. In the recoding unsupervised framework, the model transforms points from an obser-

vation space to an output space, and the output distribution is compared either to a reference distribution or to a distribution derived from the output distribution. An example is independent component analysis (*ICA*) [11], a method that discovers a representation of vector-valued observations in which the statistical dependence among the vector elements in the output space is minimized. With ICA, the model *demixes* observation vectors and the output distribution is compared against a factorial distribution which is derived either from assumptions about the distribution (e.g., supergaussian) or from a factorization of the output distribution. Other examples within the recoding framework are projection methods such as projection pursuit (e.g., [14]) and principal component analysis. In each case we have described for the unsupervised learning of a model, the objective is to bring two probability distributions—one or both of which is produced by the model—into alignment. To improve the model, we need to define a measure of the discrepancy between the two distributions, and to know how the model parameters influence the discrepancy.

One natural approach is to use outputs from the model to construct a *probability density estimator* (*PDE*). The primary disadvantage of such an approach is that the accuracy of the learning procedure depends highly on the quality of the PDE. PDEs face the bias-variance trade-off. For the learning of generative models, *maximum likelihood* (*ML*) is a popular approach that avoids PDEs. In an ML approach, the model's generative distribution is expressed analytically, which makes it straightforward to evaluate the posterior, $p(\text{data} \mid \text{model})$, and therefore, to adjust the model parameters to maximize the likelihood of the data being generated by the model. This limits the ML approach to models that have tractable posteriors, true only of the simplest models [1, 6, 9].

We describe an approach which, like ML, avoids the construction of an explicit PDE, yet does so without requiring an analytic expression for the posterior. Our approach, which we call a *sample-based method*, assumes a set of samples from each distribution and proposes an error measure of the disagreement defined directly in terms of the samples. Thus, a second set of samples drawn from the model serves in place of a PDE or an analytic expression of the model's density. The sample-based method is inspired by the theory of electric fields, which describes the interactions among charged particles. For more details on the metaphor, see [10].

In this paper, we prove that our approach converges to the optimal solution as the sample size goes to infinity, assuming an unconstrained (maximally flexible) model. We also prove that the expected solution of our approach is the ML solution in a generative context. We present empirical results showing that the sample-based approach works for both linear and nonlinear models.

## 2   The Method

Consider a model to be learned, $f_w$, parameterized by weights $w$. The model maps an input vector, $z^i$, indexed by $i$, to an output vector $x^i = f_w(z^i)$. The model inputs are sampled from a distribution $p_z(.)$, and the learning procedure calls for adjusting the model such that the output distribution, $p_x(.)$, comes to match a target distribution, $p_y(.)$. For unsupervised recoding models, $z^i$ is an observation, $x^i$ is the transformed representation of $z^i$, and $p_y(.)$ specifies the desired code properties. For unsupervised generative models, $p_z(.)$ is fixed and $p_y(.)$ is the distribution of observations.

**The Sample-based Method: The Intuitive Story**

Assume that we have data points sampled from two different distributions, labeled "−" and "+" (Figure 1). The sample-based error measure specifies how samples should be moved so that the two distributions are brought into alignment. In the figure, samples from the lower left and upper right corners must be moved to the upper left and lower right corners. Our goal is to establish an explicit correspondence between each "−" sample and each "+" sample. Toward this end, our sample-based method utilizes on mass interactions among the samples, by introducing a repelling force between samples from the same distribution and an attractive force between samples from different distributions, and allowing the samples to move according to these forces.

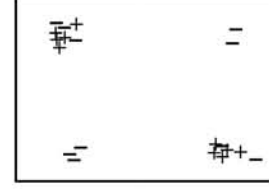

**Figure 1**

**The Sample-based Method: The Formal Presentation**

In conceiving of the problem in terms of samples that attract and repel one another, it is natural to think in terms of physical interactions among charged particles. Consider a set of positively charged particles at locations denoted by $x^i$, $i = 1...N_x$, and a set of negatively charged particles at locations denoted by $y^j$, $j = 1...N_y$. The particles correspond to data samples from two distributions. The interaction among particles is characterized by the Coulomb energy, $E$:

$$E = \frac{1}{2}\left( \frac{1}{N_x^2} \sum_{i=1}^{N_x}\sum_{k=1}^{N_x} \Gamma\left(x^i, x^k\right) - \frac{2}{N_y N_x} \sum_{i=1}^{N_x}\sum_{j=1}^{N_y} \Gamma\left(x^i, y^j\right) + \frac{1}{N_y^2} \sum_{k=1}^{N_y}\sum_{j=1}^{N_y} \Gamma\left(y^k, y^j\right) \right),$$

where $\Gamma(a, b)$ is a distance measure—Green's function—which results in nearby particles having a strong influence on the energy, but distant particles having only a weak influence. Green's function is defined as $\Gamma(a, b) = c(d) \,/\, \|a - b\|^{d-2}$, where $d$ is the dimensionality of the space, $c(d)$ is a constant only depending on $d$, and $\|.\|$ denotes the Euclidean distance. For $d = 2$, $\Gamma(a, b) = k \ln\left(\|a - b\|\right)$.

The Coulomb energy is low when negative and positive particles are near one another, positive particles are far from one another, and negative particles are far from one another. This is exactly the state we would like to achieve for our two distributions of samples: bringing the two distributions into alignment without collapsing either distribution into a trivial form. Consequently, our sample-based method proposes using the Coulomb energy as an objective function to be minimized.

The gradient of $E$ with respect to a sample's location is readily computed (it is the force acting on that sample), and this gradient can be chained with the Jacobian of the location with respect to the model parameters $w$ to obtain a gradient-based update rule: $\Delta w = -\epsilon \nabla_w E = -\epsilon \left( \frac{1}{N_x} \sum_{k=1}^{N_x} \left(\frac{\partial x^k}{\partial w}\right)^T \nabla_{x^k} \Phi\left(x^k\right) - \frac{1}{N_y} \sum_{k=1}^{N_y} \left(\frac{\partial y^k}{\partial w}\right)^T \nabla_{y^k} \Phi\left(y^k\right) \right)$, where $\epsilon$ is a step size, $\Phi(a) := N_x^{-1} \sum_{i=1}^{N_x} \Gamma(a, x^i) - N_y^{-1} \sum_{j=1}^{N_y} \Gamma(a, y^j)$ is the potential with $N_a^{-1} \nabla_a \Phi(a) = \nabla_a E$, $T$ is the transposition and $a = x^k$ or $y^k$. Here $\partial x^k / \partial w$ is the Jacobian of $f_w(z^k)$ and the time derivative of $x^k$ is $\dot{x}^k = \dot{f}_w(z^k) = -\nabla \Phi(x^k)$. If $y^k$ depends on $w$ then $y^k$–notation is analogous else $\partial y^k / \partial w$ is the zero matrix.

There turns out to be an advantage to using Green's function as the particle interactions basis over other possibilities, e.g., a Gaussian function (e.g., [12, 13, 3]).

The advantage stems from the fact that with Green's function, the force between two nearby points goes to infinity as the points are pushed together, whereas with the Gaussian, the force goes to zero. Consequently, without Green's function, one might expect local optima in which clusters of points collapse onto a single location. Empirically, simulations confirmed this conjecture.

**Proof: Correctness of the Update Rule**

As the numbers of samples $N_x$ and $N_y$ go to infinity, $\Phi$ can be expressed as $\Phi(a) = \int \rho(b)\, \Gamma(a,b)\, db$, where $\rho(b) := p_x(b) - p_y(b)$. Our sample-based method moves data points, but by moving data points, the method implicitly alters the probability density which gave rise to the data. The relation between the movement of data points and the change in the density can be expressed using an operator from vector analysis, the *divergence*. The divergence at a location $a$ is the number of data points moving *out* of a volume surrounding $a$ minus the number of data points moving *in* to the same volume. Thus, the negative divergence of movements at $a$ gives the density change at $a$. The movement of data points is given by $-\nabla\Phi(a)$. We get $\dot{\rho}(a) = \dot{p}_x(a) - \dot{p}_y(a) = -\mathrm{div}\,(-\nabla\Phi(a))$. For Cartesian (orthogonal) coordinates the divergence *div* of a vector field $V$ at $a$ is defined as $\mathrm{div}\,(V(a)) := \sum_{l=1}^{d} \partial V_l(a)/\partial a_l$. The Laplace operator $\triangle$ of a scalar function $A$ is defined as $\triangle A(a) := \mathrm{div}\,(\nabla A(a)) = \sum_{l=1}^{d} \partial^2 A(a)/\partial a_l^2$. The Laplace operator allows an important characterization of Green's function: $\triangle_a \Gamma(a,b) = -\delta(a-b)$, where $\delta$ is the Dirac delta function. This characterization gives $\triangle\Phi(a) = -\rho(a)$.

$$\dot{\rho}(a) = \mu(a)\,\mathrm{div}\,(\nabla\Phi(a)) = \mu(a)\,\triangle\Phi(a) = -\mu(a)\,\rho(a)\,, \quad \mu(a) \geq \mu_0 > 0\,,$$

where $\mu(a)$ gives the effectiveness of the algorithm in moving a sample at $a$. We get $\rho(a,t) = \rho(a,0)\,\exp(-\mu(a)\,t)$. For the integrated squared error (ISE) of the two distributions we obtain

$$ISE(t) = \int (\rho(a,t))^2\, da \leq \exp(-\mu_0\, t) \int (\rho(a,0))^2\, da = \exp(-\mu_0\, t)\, ISE(0)\,,$$

where $ISE(0)$ is independent of $t$. Thus, the ISE between the two distributions is guaranteed to decrease during learning, when the sample size goes to infinity.

**Proof: Expected Generative Solution is ML Solution**

In the case of a generative model which has no constraints (i.e., can model any distribution), the maximum likelihood solution will have distribution $p_x(a) = \frac{1}{N_y}\sum_{j=1}^{N_y} \delta(y^j - a)$, i.e., the model will produce only the observations and all of them with equal probability. For this case, we show that our sample-based method will yield the same solution in expectation as ML.

The sample-based method converges to a local minimum of the energy, where $\langle \nabla_a \Phi(a)\rangle_x = 0$ for all $a$, where $\langle .\rangle_x$ is the expectation over model output. Equivalently, $\langle \nabla_a \Gamma(a,x)\rangle_x - \frac{1}{N_y}\sum_{j=1}^{N_y} \nabla_a \Gamma(a,y^j) = 0$ or

$$\langle \nabla_a \Gamma(a,x)\rangle_x = \int p_x(x)\, \nabla_a \Gamma(a,x)\, dx = \frac{1}{N_y}\sum_{j=1}^{N_y} \nabla_a \Gamma(a,y^j)\,.$$

Because this equation holds for all $a$, we obtain $p_x(a) = \frac{1}{N_y}\sum_{j=1}^{N_y} \delta(y^j - a)$, which is the ML solution. Thus, the sample-based method can be viewed as an approximation to ML which gets more exact as the number of samples goes to infinity.

## 3 Experiments

We illustrate the sample-based approach for two common unsupervised learning problems: MoG and ICA. In both cases, we demonstrate that the sample-based approach works in the linear case. We also consider a nonlinear case to illustrate the power of the sample-based approach.

### Mixture of Gaussians

In this generative model framework, $m$ denotes a mixture component which is chosen with probability $v_m$ from $M$ components, and has associated model parameters $w_m = (\Omega_m, \mu_m)$. In the standard MoG model, given a choice of component $m$, the (linear) model output is obtained by $x^i = f_{w_m}(z^i) = \Omega_m z^i + \mu_m$, where $z^i$ is drawn from the Gaussian distribution with zero mean and identity covariance matrix. For a nonlinear mixture model, we used a 3-layer sigmoidal neural network for $f_{w_m}(z^i)$. An update rule for $v_m$ can be derived for our approach: $\Delta v_m = -\epsilon_v \sum_{i=1}^{N_x} \left(z^i\right)^T \frac{\partial z^i}{\partial x^i} \dot{x}^i$, where $\epsilon_v$ is a step size and $\sum_{m=1}^{M} v_m = 1$ is enforced.

We trained a linear MoG model with the standard expected maximization (EM) algorithm (using code from [5]) and a linear and a nonlinear MoG with our sample-based approach. A fixed training set of $N_y = 100$ samples was used for all models, and all models had $M = 10$ except one nonlinear model which had $M = 1$. In the sample-based approach, we generated 100 samples from our model (the $x^i$) following every training epoch. The nonlinear model was trained with backpropagation.

Figure 2 shows the results. The linear ML model is better than the sample-based model. That is not surprising because ML computes the model probability values analytically (the posterior is tractable) and our algorithm uses only samples to approximate the model probability values. We used only 100 model samples in each epoch and the linear sample-based model found an acceptable solution and is not much worse than the ML model. The nonlinear models fit better the true ring-like distribution and do not suffer from sharp corners and edges.

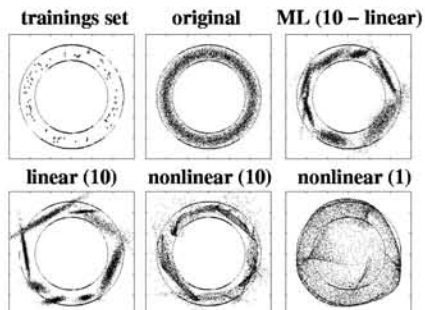

trainings set · original · ML (10 – linear)

linear (10) · nonlinear (10) · nonlinear (1)

Figure 2: (upper panel, left to right) training samples chosen from a ring density, a larger sample from this density, the solutions obtained from the linear model trained with EM; (lower panels) models trained with the sample-based method (left to right): linear model, nonlinear model, nonlinear model with one component.

### Independent Component Analysis

With a *recoding* model we tried to demix subgaussian source distributions where each has supergaussian modes. Most ICA methods are not able to demix subgaussian sources. Figure 3 shows the results, which are nearly perfect. The ideal result is a scaled and permuted identity matrix when the mixing and demixing matrices are multiplied. For more details see [10].

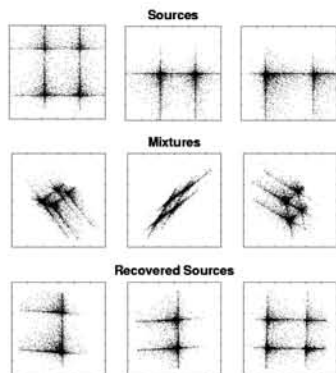

Figure 3: For a three-dimensional linear mixture projections of sources (first row), mixtures (second row), and sources recovered by our approach (third row) on a two-dimensional plane are shown.

The demixing matrix multiplied with the mixing matrix yields:

| | | |
|---|---|---|
| -0.0017 | 0.0010 | **0.2523** |
| -0.0014 | **0.1850** | -0.0101 |
| **-0.1755** | 0.0003 | 0.0053 |

In a second experiment, we tried to recover sources from two nonlinear mixings. This problem is impossible for standard ICA methods because they are designed for linear mixings. The result is shown in Figure 4. An exact demixing cannot be expected, because nonlinear ICA has no unique solution. For more details see [10].

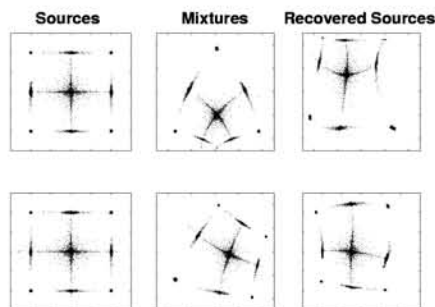

Figure 4: For two two-dimensional nonlinear mixing functions— upper row, $(z + a)^2$, and lower row, $\sqrt{z + a}$, with complex variable $z$—the sources, mixtures, and recovered sources. The mixing function is not completely inverted but the sources are recovered recognizable.

## 4    Discussion

Although our sample-based approach is intuitively straightforward, its implementation has two drawbacks: (1) One has to be cautious of samples that are close together, because they lead to unbounded gradients; and (2) all samples must be considered when computing the force on a data point, which makes the approach computation intensive. However, in [10, 7] approximations are proposed that reduce the computational complexity of the approach.

In this paper, we have presented simulations showing the generality and power of our sample-based approach to unsupervised learning problems, and have also proven two important properties of the approach: (1) With certain assumptions, the approach will find the correct solution. (2) With an unconstrained model, the expected solution of our approach is the ML solution. In conclusion, our sample-based approach can be applied to unsupervised learning of complex models where ML does not work and our method avoids the drawbacks of PDE approaches.

## Acknowledgments

We thank Geoffrey Hinton for inspirational suggestions regarding this work. The work was supported by the *Deutsche Forschungsgemeinschaft* (Ho 1749/1-1), McDonnell-Pew award 97-18, and NSF award IBN-9873492.

## References

[1] P. Dayan, G. E. Hinton, R. M. Neal, and R. S. Zemel. The Helmholtz machine. *Neural Computation*, 7(5):889–904, 1995.

[2] R. O. Duda and P. E. Hart. *Pattern Classification and Scene Analysis*. Wiley, 1973.

[3] D. Erdogmus and J. C. Principe. Comparision of entropy and mean square error criteria in adaptive system training using higher order statistics. In P. Pajunen and J. Karhunen, editors, *Proceedings of the Second International Workshop on Independent Component Analysis and Blind Signal Separation, Helsinki, Finland*, pages 75–80. Otamedia, Espoo, Finland, ISBN: 951-22-5017-9, 2000.

[4] B. S. Everitt. *An introduction to latent variable models*. Chapman and Hall, 1984.

[5] Z. Ghahramani and G. E. Hinton. The EM algorithm for mixtures of factor analyzers. Technical Report CRG-TR-96-1, University of Toronto, Dept. of Comp. Science, 1996.

[6] Z. Ghahramani and G. E. Hinton. Hierachical non-linear factor analysis and topographic maps. In M. I. Jordan, M. J. Kearns, and S. A. Solla, editors, *Advances in Neural Information Processing Systems 10*, pages 486–492. MIT Press, 1998.

[7] A. Gray and A. W. Moore. 'N-body' problems in statistical learning. In T. K. Leen, T. Dietterich, and V. Tresp, editors, *Advances in Neural Information Processing Systems 13*, 2001. In this proceeding.

[8] G. E. Hinton and T. J. Sejnowski. Learning and relearning in Boltzmann machines. In *Parallel Distributed Processing*, volume 1, pages 282–317. MIT Press, 1986.

[9] G. E. Hinton and T. J. Sejnowski. Introduction. In G. E. Hinton and T. J. Sejnowski, editors, *Unsupervised Learning: Foundations of Neural Computation*, pages VII–XVI. The MIT Press, Cambridge, MA, London, England, 1999.

[10] S. Hochreiter and M. C. Mozer. An electric field approach to independent component analysis. In P. Pajunen and J. Karhunen, editors, *Proceedings of the Second International Workshop on Independent Component Analysis and Blind Signal Separation, Helsinki, Finland*, pages 45–50. Otamedia, Finland, ISBN: 951-22-5017-9, 2000.

[11] A. Hyvärinen. Survey on independent component analysis. *Neural Computing Surveys*, 2:94–128, 1999.

[12] G. C. Marques and L. B. Almeida. Separation of nonlinear mixtures using pattern repulsion. In J.-F. Cardoso, C. Jutten, and P. Loubaton, editors, *Proceedings of the First International Workshop on Independent Component Analysis and Signal Separation, Aussois, France*, pages 277–282, 1999.

[13] J. C. Principe and D. Xu. Information-theoretic learning using Renyi's quadratic entropy. In J.-F. Cardoso, C. Jutten, and P. Loubaton, editors, *Proceedings of the First International Workshop on Independent Component Analysis and Signal Separation, Aussois, France*, pages 407–412, 1999.

[14] Y. Zhao and C. G. Atkeson. Implementing projection pursuit learning. *IEEE Transactions on Neural Networks*, 7(2):362–373, 1996.